# How fast to work: Response vigor, motivation and tonic dopamine

**Yael Niv**[1,2]    **Nathaniel D. Daw**[2]    **Peter Dayan**[2]
[1]ICNC, Hebrew University, Jerusalem    [2]Gatsby Computational Neuroscience Unit, UCL
`yaelniv@alice.nc.huji.ac.il`  `{daw,dayan}@gatsby.ucl.ac.uk`

## Abstract

Reinforcement learning models have long promised to unify computational, psychological and neural accounts of appetitively conditioned behavior. However, the bulk of data on animal conditioning comes from free-operant experiments measuring how fast animals will work for reinforcement. Existing reinforcement learning (RL) models are silent about these tasks, because they lack any notion of *vigor*. They thus fail to address the simple observation that hungrier animals will work harder for food, as well as stranger facts such as their sometimes greater productivity even when working for irrelevant outcomes such as water. Here, we develop an RL framework for free-operant behavior, suggesting that subjects choose how vigorously to perform selected actions by optimally balancing the costs and benefits of quick responding. Motivational states such as hunger shift these factors, skewing the tradeoff. This accounts normatively for the effects of motivation on response rates, as well as many other classic findings. Finally, we suggest that tonic levels of dopamine may be involved in the computation linking motivational state to optimal responding, thereby explaining the complex vigor-related effects of pharmacological manipulation of dopamine.

## 1   Introduction

A banal, but nonetheless valid, behaviorist observation is that hungry animals work harder to get food [1]. However, associated with this observation are two stranger experimental facts and a large theoretical failing. The first weird fact is that hungry animals will in some circumstances work more vigorously even for motivationally irrelevant outcomes such as water [2, 3], which seems highly counterintuitive. Second, contrary to the emphasis theoretical accounts have placed on the effects of dopamine (DA) on learning to choose between actions, the most overt behavioral effects of DA interventions are similar swings in undirected vigor [4], at least part of which appear immediately, without learning [5]. Finally, computational theories fail to deliver on the close link they trumpet between DA, behavior, and reinforcement learning (RL; *eg* [6]), as they do not address the whole experimental paradigm of *free-operant* tasks [7], whence hail those and many other results.

Rather than the standard RL problem of discrete choices between alternatives at prespecified timesteps [8], free-operant experiments investigate tasks in which subjects pace their own responding (typically on a lever or other manipulandum). The primary choice in these tasks is of how rapidly/vigorously to behave, rather than what behavior to choose (as typically only one relevant action is available). RL models are silent about these aspects, and thus fail to offer a principled understanding of the policies selected by the animals.

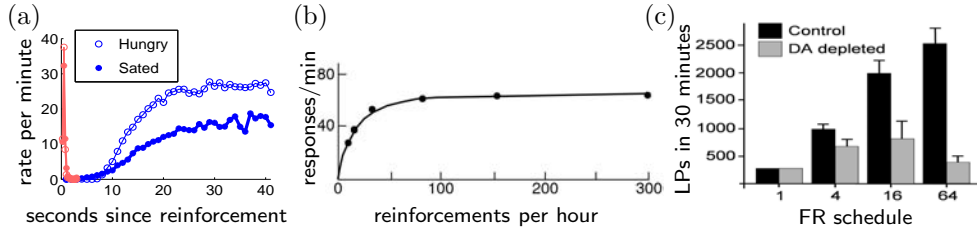

Figure 1: (a) Leverpress (blue, right) and consummatory nose poke (red, left) response rates of rats leverpressing for food on a modified RI30 schedule. Hungry rats (open circles) clearly press the lever at a higher rate than sated rats (filled circles). Data from [11], averaged over 19 rats in each group. (b) The relationship between rate of responding and rate of reinforcement (reciprocal of the interval) on an RI schedule, is hyperbolic (of the form $y = B \cdot x/(x + x_0)$). This is an instantiation of Herrnstein's matching law for one response (adapted from [9]). (c) Total number of leverpresses per session averaged over five 30 minute sessions by rats pressing for food on different FR schedules. Rats with nucleus accumbens 6-OHDA dopamine lesions (gray) press significantly less than control rats (black), with the difference larger for higher ratio requirements. Adapted from [12].

Here, we address these issues by constructing an RL account of behavior rates in free-operant settings (Sections 2,3). We consider optimal control in a continuous-time Markov Decision Process (MDP), in which agents must choose *both* an action and the latency with which to emit it (*ie* how vigorously, or at what instantaneous rate to perform it). Our model treats response vigor as being determined normatively, as the outcome of a battle between the cost of behaving more expeditiously and the benefit of achieving desirable outcomes more quickly. We show that this simple, normative framework captures many classic features of animal behavior that are obscure in our and others' earlier treatments (Section 4). These include the characteristic time-dependent profiles of response rates on tasks with different payoff scheduling [7], the hyperbolic relationship between response rate and payoff [9], and the difference in response rates between tasks in which reinforcements are allocated based on the number of responses emitted and those allocating reinforcements based on the passage of time [10].

A key feature of this model is that response rates are strongly dependent on the expected *average reward rate*, because this determines the opportunity cost of sloth. By influencing the value of reinforcers — and through this, the average reward rate — motivational states such as hunger influence the output response latencies (and not only response choice). Thus, in our model, hungry animals should *optimally* also work harder for water, since in typical circumstances, this should allow them to return more quickly to working for food. Further, we identify *tonic levels of dopamine* with the representation of average reward rate, and thereby suggest an account of a wealth of experiments showing that DA influences response vigor [4, 5], thus complementing existing ideas about the role of phasic DA signals in learned action selection (Section 5).

## 2   Free-operant behavior

We consider the free-operant scenario common in experimental psychology, in which an animal is placed in an experimental chamber, and can choose freely which actions to emit and when. Most actions have no programmed consequences; however, one action (*eg* leverpressing; LP) is rewarded with food (which falls into a food magazine) according to an experimenter-determined *schedule* of reinforcement. Food delivery makes a characteristic sound, signalling its availability for harvesting via a nose poke (NP) into the magazine.

The schedule of reinforcement defines the (possibly stochastic) relationship between the delivery of a reward and one or both of (a) the *number* of LPs, and (b) the *time* since the last reward was delivered. In common use are fixed-ratio (FR) schedules, in which a fixed number of LPs is required to obtain a reinforcer; random-ratio (RR) schedules, in which each LP has a constant probability of being reinforced; and random interval (RI) schedules, in which the first LP after an (exponentially distributed) interval of time has elapsed, is reinforced. Schedules are often labelled by their type and a parameter, so RI30 is a random interval schedule with the exponential waiting time having a mean of 30 seconds [7].

Different schedules induce different patterns of responding [7]. Fig 1a shows response metrics from rats leverpressing on an RI30 schedule. Leverpressing builds up to a relatively constant rate following a rather long pause after gaining each reward, during which the food is consumed. Hungry rats leverpress more vigorously than sated ones. A similar overall pattern is also characteristic of responding on RR schedules. Figure 1b shows the total number of LP responses in a 30 minute session for different interval schedules. The hyperbolic relationship between the reward rate (the inverse of the interval) and the response rate is a classic hallmark of free operant behavior [9].

## 3 The model

We model a free-operant task as a continuous MDP. Based on its state, the agent chooses both an action $(a)$, and a latency $(\tau)$ at which to emit it. After time $\tau$ elapsed, the action is completed, the agent receives rewards and incurs costs associated with its choice, and then selects a new $(a, \tau)$ pair based on its new state. We define three possible actions $a \in \{\texttt{LP}, \texttt{NP}, \texttt{other}\}$, where we take $a = \texttt{other}$ to include the various miscellaneous behaviors such as grooming, rearing, and sniffing which animals typically perform during the experiment. For simplicity we consider unit actions, with the latency $\tau$ related to the vigor with which this unit is performed. To account for consumption time (which is non-negligible [11, 13]), if the agent nose-pokes and food is available, a predefined time $t_{\text{eat}}$ passes before the next decision point (and the next state) is reached.

Crucially, performing actions incurs costs as well as potentially gains rewards. Following Staddon [14], we assume one part of the cost of an action to be *proportional to the vigor* of its execution, *ie* inversely proportional to $\tau$. The constant of proportionality $K_v$ depends on both the previous and the current action, since switching between different action types can require travel between different parts of the experimental chamber (say, the magazine to the lever), and can thus be more costly. Each action also incurs a fixed 'internal' reward or cost of $\rho(a)$ per unit, typically with $\texttt{other}$ being rewarding. The reinforcement schedule defines the probability of reward delivery for each state-action-latency triplet. An available reward can be harvested by $a = \texttt{NP}$ into the magazine, and we assume that the thereby obtained subjective utility $U(r)$ of the food reward is *motivation-dependent*, such that food is worth more to a hungry animal than to a sated one.

We consider the simplified case of a state space comprised of all the parameters relevant to the task. Specifically, the state space includes the identity of the previous action, an indicator as to whether a reward is available in the food magazine, and, as necessary, the number of LPs since the previous reinforcement (for FR) or the elapsed time since the previous LP (for RI). The transitions between the states $P(S'|S, a, \tau)$ and the reward function $P_r(S, a, \tau)$ are defined by the dynamics of the schedule of reinforcement, and all rewards and costs are harvested at state transitions and considered as point events. In the following we treat the problem of optimising a policy (which action to take and with what latency, given the state) in order to maximize the average rate of return (rewards minus costs per time). An exponentially discounted model gives the same qualitative results.

In the average reward case [15, 16], the Bellman equation for the long-term differential (or

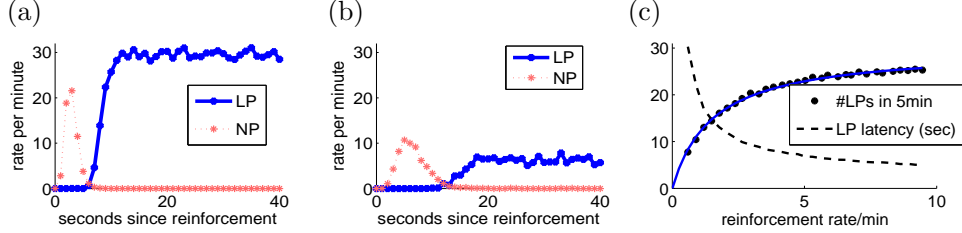

Figure 2: Data generated by the model captures the essence of the behavioral data: Lever-press (solid blue; circles) and nose poke (dashed red; stars) response rates on (a) an RR10 schedule and (b) a matched (yoked) RI schedule show constant LP rates which are higher for the ratio schedule. (c) The relationship between the total number of responses (circles) and rate of reinforcement is hyperbolic (solid line: hyperbolic curve fit). The mean latency to leverpress (dashed line) decreases as the rate of reinforcement increases.

average-adjusted) value of state $S$ is:

$$V^*(S) = \max_{a,\tau} \left\{ \rho(a) - \frac{K_v(a_{prev}, a)}{\tau} + U(r)P_r(S, a, \tau) - \tau \cdot \overline{r} + \int dS' P(S'|S, a, \tau)V^*(S') \right\} \quad (1)$$

where $\overline{r}$ is the long term average reward rate (whose subtraction from the value quantifies the opportunity cost of delay). Building on ideas from [16], we suggest that the average reward rate is reported by tonic (baseline) levels of dopamine (and *not* serotonin [16]) in basal ganglia structures relevant for action selection, and that changes in tonic DA (*eg* as a result of pharmacological interventions) would thus alter the assumed average reward rate.

In this paper, we eschew learning, and examine the steady state behavior that arises when actions are chosen stochastically (via the so-called softmax or Boltzmann distribution) from the *optimal* one-step look-ahead model-based $Q(S, a, \tau)$ state-action-latency values. For ratio schedules, the simple transition structure of the task allows the Bellman equation to be solved analytically to determine the $Q$ values. For interval schedules, we use average-reward value iteration [15] with time discretized at a resolution of 100ms. For simulations (*eg* of dopaminergic manipulations) where $\overline{r}$ was assumed to change independent of any change in the task contingencies, we used value iteration to find values approximately satisfying the Bellman equation (which is no longer exactly solvable). Our overriding aim is to replicate basic aspects of free operant behavior qualitatively, in order to understand the normative foundations of response vigor. We do not fit the parameters of the model to experimental data in a quantitative way, and the results we describe below are general, robust, characteristics of the model.

## 4 Results

Fig 2a depicts the behavior of our model on an RR10 schedule. In rough accordance with the behavior displayed by animals (which is similar to that shown in Fig 1a), the LP rate is constant over time, bar a pause for consumption. Fig 2b depicts the model's behavior in a *yoked* random interval schedule, in which the intervals between rewards were set to match exactly the intervals obtained by the agent trained on the ratio schedule in Fig 2a. The response rate is again constant over time, but it is also considerably *lower* than that in the corresponding RR schedule, although the external reward density is similar. This phenomenon has also been observed experimentally, and although the apparent anomaly has been much discussed in the associative learning literature, its explanation is not fully resolved [10]. Our model suggests that it is the result of an optimal cost/benefit tradeoff.

We can analyse this difference by considering the $Q$ values for leverpressing at different

latencies in random schedules

$$Q(S_{nr}, \text{LP}, \tau) = \rho(\text{LP}) - \frac{K_v(\text{LP}, \text{LP})}{\tau} - \tau \cdot \bar{r} + P(S_r|\tau)V^*(S_r) + [1 - P(S_r|\tau)]V^*(S_{nr}) \quad (2)$$

where we are looking at consecutive leverpresses in the absence of available reward, and $S_r$ and $S_{nr}$ designate the states in which a reward is or is not available in the magazine, respectively. In ratio schedules, since $P(S_r|\tau)$ is independent of $\tau$, the optimizing latency is $\tau_{\text{LP}}^* = \sqrt{K_v(\text{LP}, \text{LP})/\bar{r}}$, its inverse defining the optimal rate of leverpressing. In interval schedules, however, $P(S_r|\tau) = 1 - exp\{-\tau/T\}$ where $T$ is the schedule interval. Taking the derivative of eq. (2) we find that the optimal latency to leverpress $\tau_{\text{LP}}^*$ satisfies $K_v(\text{LP}, \text{LP})/\tau_{\text{LP}}^{*2} - \bar{r} + (1/T)[V^*(S_r) - V^*(S_{nr})] \cdot exp\{-\tau_{\text{LP}}^*/T\} = 0$. Although no longer analytically solvable, it is easily seen that this latency will always be longer than that found above for ratio schedules. Intuitively, since longer inter-response intervals increase the probability of reward per press in interval schedules but not in ratio schedules, the optimal leverpressing rate is lower in the former than in the latter.

Fig 2c shows the average number of LPs in a 5 minute session for different interval schedules. This 'molar' measure of rate shows the well documented hyperbolic relationship (*cf* Fig 1b). On the 'molecular' level of single action choices, the mean latency $\langle \tau_{\text{LP}} \rangle$ between consecutive LPs decreases as the probability of reinforcement increases. This measure of response vigor is actually more accurate than the overall response measure, as it is not contaminated by competition with other actions, or confounded with the number of reinforcers per session for different schedules (and the time forgone when consuming them). For this reason, although we (correctly; see [13]) predict that inter-response latency should slow for higher ratio requirements, raw LP counts can actually increase, as in Fig. 1c, probably due to fewer rewards and less time spent eating [13].

## 5   Drive and dopamine

Having provided a qualitative account of the basic patterns of free operant rates of behavior, we turn to the main theoretical conundrum — the effects of drive and DA manipulations on response vigor. The key to understanding these is the role that the average reward $\bar{r}$ plays in the tradeoffs determining optimal response vigor. In effect, the average expected reward per unit time quantifies the opportunity cost for doing nothing (and receiving no reward) for that time; its increase thus produces general pressure for faster work. A direct consequence of making the agent hungrier is that the subjective utility of food is enhanced. This will have interrelated effects on the optimal average reward $\bar{r}$, the optimal values $V^*$, and the resultant optimal action choices and vigors. Notably, so long as the policy obtains food, its average reward rate will *increase*.

Consider a fixed or random ratio schedule. The increase in $\bar{r}$ will increase the optimal LP rate $1/\tau_{\text{LP}}^* = \sqrt{\bar{r}/K_v(\text{LP}, \text{LP})}$, as the higher reward utility offsets higher procurement costs. Importantly, because the optimal $\tau^*$ has a similar dependence on $\bar{r}$ even for actions irrelevant to obtaining food, they also become more vigorous. The explanation of this effect is presented graphically in Fig 3e. The higher $\bar{r}$ increases the cost of sloth, since every $\tau$ time without reward forgoes an expected $(\tau \cdot \bar{r})$ mean reward. Higher average rewards penalize late actions more than they do early ones, thus tilting action selection toward faster behavior, for *all* pre-potent actions. Essentially, hunger encourages the agent to complete irrelevant actions faster, in order to be able to resume leverpressing more quickly.

For other schedules, the same effects generally hold (although the analytical reasoning is complicated by the fact that the optimal latencies may in these cases depend not only on the new average reward but also on the new values $V^*$). Fig 3a shows simulated responding on an RI25 schedule in which the internal reward for the food-irrelevant action `other` has been set high enough to warrant non-negligible base responding. Fig 3b shows that when

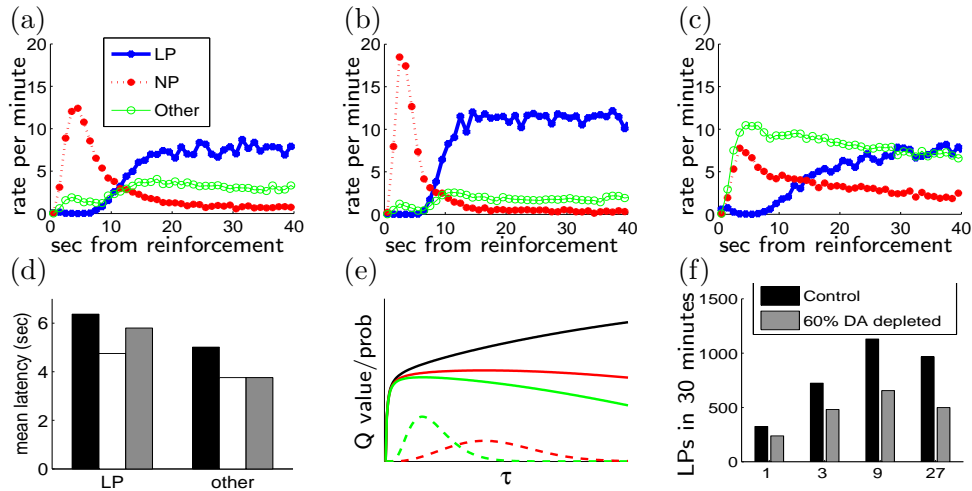

Figure 3: The effects of drive on response rates. (a) Responding on a RI25 schedule, with high internal rewards (0.35) for $a = \texttt{other}$ (open circles). (b) The effects of hunger: $U(r)$ was changed from 10 to 15. (c) The effect of an irrelevant drive (hungry animals lever-pressing for water rewards): $\bar{r}$ was increased by $4\%$ compared to (a). (d) Mean latencies to responding $\langle\tau\rangle$ for LP and $\texttt{other}$ in baseline (a; black), increased hunger (b; white) and irrelevant drive (c; gray). (e) Q values for leverpressing at different latencies $\tau$. In black (top) are the unadjusted Q values, before subtracting $(\tau \cdot \bar{r})$. In red (middle, solid) and green (bottom, solid) are the values adjusted for two different average reward rates. The higher reward rate penalizes late actions more, thereby causing *faster* responding, as shown by the corresponding softmaxed action probability curves (dashed). (f) Simulation of DA depletion: overall leverpress count over 30 minute sessions (each bar averaging 15 sessions), for different FR requirements (bottom). In black is the control condition, and in gray is simulated DA depletion, attained by lowering $\bar{r}$ by $60\%$. The effects of the depletion seem more pronounced in higher schedules (compare to Fig 1c), but this actually results from the interaction with the number of rewards attained (see text).

the utility of food is increased by $50\%$, the agent chooses to leverpress more, at the expense of $\texttt{other}$ actions. This illustrates the 'directing' effect of motivation, by which the agent is directed more forcefully toward the motivationally relevant action [17]. Furthermore, the second, 'driving' effect, by which motivation increases vigor globally [17], is illustrated in Fig 3d which shows that, in fact, the latency to both actions has *decreased*. Thus, although selected less often, when $\texttt{other}$ is selected, it is performed more vigorously than it was when the agent was sated.

This general drive effect can be better isolated if we examine hungry agents leverpressing for water (rather than food), without competition from actions for food. We can view our leverpressing MDP as a portion of a larger one, which also includes (for instance) occasional opportunities for visits to a home cage where food is available. Without explicitly specifying all this extra structure, a good approximation is to take hunger as again causing an increase in the global rate of reinforcement $\bar{r}$, reflecting the increase in the utility of food received elsewhere. Fig 3c shows the effects on responding on an interval schedule, of estimating the average reward rate to be $4\%$ higher than in Fig 3a, and deriving new Q values from the previous $V^*$ with this new $\bar{r}$ as illustrated in Fig 3e. As above, the adjusted vigors of all behaviors are faster (Fig 3d, gray bars), as a result of the higher 'drive'.

How do these drive effects relate to dopamine? Pharmacological and lesion studies show that enhancing DA levels (through agonists such as amphetamine) increases general activity

[5, 18, 19], while depleting or antagonising DA causes a general slowing of responding (*eg* [4]). Fig. 1c is representative of a host of results from the lab of Salamone [4, 12] which show that lower levels of DA in the nucleus accumbens (a structure in the basal ganglia implicated in action selection) result in lower response rates. This effect seems more pronounced in higher fixed-ratio schedules, those requiring more work per reinforcer. As a result of this apparent dependence on the response requirement, Salamone and his colleagues have hypothesized that DA enables animals to overcome higher work demands.

We suggest that tonic levels of DA represent the average reward rate (a role tentatively proposed for *serotonin* in [16]). Thus a higher tonic level of DA represents a situation akin to higher drive, in which behavior is more vigorous, and lower tonic levels of DA cause a general slowing of behavior. Fig. 3f shows the simulated response counts for different FR schedules in two conditions. The control condition is the standard model described above; DA depletion was modeled by decreasing tonic DA levels (and therefore $\bar{r}$) to $40\%$ of their original levels. The results match the data in Fig. 1c. Here, the apparently small effect on the number of LPs for low ratio schedules actually arises because of the large amount of time spent eating. Thus, according to the model DA is not really allowing animals to cope with higher work requirements, but rather is important for optimal choice of vigor at any work requirement, with the slowing effect of DA depletion more prominent (in the crude measure of LPs per session) when more time is spent leverpressing.

## 6 Discussion

The present model brings the computational machinery and neural grounding of RL models fully into contact with the vast reservoir of data from free-operant tasks. Classic quantitative accounts of operant behavior (such as Herrnstein's matching law [9], and variations such as melioration) lack RL's normative grounding in sound control theory, and tend instead toward descriptive curve-fitting. Most of these theories do not address that fine scale (molecular) structure of behavior, and instead concentrate on fairly crude molar measures such as total number of leverpresses over long durations. In addition to the normative starting point it offers for investigations of response vigor, our theory provides a relatively fine scalpel for dissecting the temporal details of behavior, such as the distributions of inter-response intervals at particular state transitions. There is thus great scope for revealing re-analyses of many existing data sets. In particular, the effects of generalized drive have proved mixed and complex [17]. Our theory suggests that studies of inter-response intervals (*eg* Fig 3d) may reveal more robust changes in vigor, uncontaminated by shifts in overall action propensity.

Response vigor and dopamine's role in controlling it have appeared in previous RL models of behavior [20, 21], but only as fairly ad-hoc bolt-ons — for instance, using repeated choices between doing nothing versus something to capture response latency. Here, these aspects are wholly integrated into the explanatory framework: optimizing response vigor is treated as itself an RL problem, with a natural dopaminergic substrate. To account for immediate (unlearned) effects of motivational or dopaminergic manipulations, the main assumption we make is that tonic levels of DA can be sensitive to predicted changes in the average reward occasioned by changes in the motivational state, and that behavioral policies are in turn immediately affected. This sensitivity would be easy to embed in a temporal-difference RL system, producing flexible adaptation of response vigor. By contrast, due to the way they cache outcome values, the action choices of such RL systems are characteristically *insensitive* to the 'directing' effects of motivational manipulations [22]. In animal behavior, 'habitual actions' (the ones associated with the DA system) are indeed motivationally insensitive for action choice, but show a direct effect of drive on vigor [23].

Our model is easy to accommodate within a framework of temporal difference (TD) learning. Thus, it naturally preserves the link between phasic DA signals and online learning

of optimal values [24]. We further elaborate this link by suggesting an additional role for tonic levels of DA in online vigor selection. A major question remains as to whether phasic responses (which are known to correlate with response latency [25]) play an additional role in determining response vigor. Further, it is pressing to reconcile the present account with our previous suggestion (based on microdialysis findings) [16] that tonic levels of DA might track average *punishment*.

The most critical avenues to develop this work will be an account of learning, and neurally and psychologically more plausible state and temporal representations. On-line value learning should be a straightforward adaptation of existing TD models of phasic DA based on the continuous-time semi-Markov setting [26]. The representation of state is more challenging — the assumption of a fully observable state space automatically appropriate for the schedule of reinforcement is not realistic. Indeed, apparently sub-optimal actions emitted by animals, *eg* engaging in excessive nose-poking even when a reward has not audibly dropped into the food magazine [11], may provide clues to this issue. Finally, it will be crucial to consider the fact that animals' decisions about vigor may translate only noisily into response times, due, for instance, to the variability of internal timing [27].

## Acknowledgments

This work was funded by the Gatsby Charitable Foundation, a Dan David fellowship (YN), the Royal Society (ND) and the EU BIBA project (ND and PD). We are grateful to Jonathan Williams for discussions on free operant behavior.

## References

[1] Dickinson A. and Balleine B.W. The role of learning in the operation of motivational systems. Steven's Handbook of Experimental Psychology Volume 3, pages 497–533. John Wiley & Sons, New York, 2002.

[2] Hull C.L. *Principles of behavior: An introduction to behavior theory*. Appleton-Century-Crofts, New York, 1943.

[3] Bélanger D. and Tétreau B. L'influence d'une motivation inappropriée sur le comportement du rat et sa fréquence cardiaque. *Can. J. of Psych.*, 15:6–14, 1961.

[4] Salamone J.D. and Correa M. Motivational views of reinforcement: implications for understanding the behavioral functions of nucleus accumbens dopamine. *Behavioural Brain Research*, 137:3–25, 2002.

[5] Ikemoto S. and Panksepp J. The role of nucleus accumbens dopamine in motivated behavior: a unifying interpretation with special reference to reward-seeking. *Brain Res. Rev.*, 31:6–41, 1999.

[6] Schultz W. Predictive reward signal of dopamine neurons. *J. Neurophys.*, 80:1–27, 1998.

[7] Domjan M. *The principles of learning and behavior*. Brooks/Cole, Pacific Grove, California, 3rd edition, 1993.

[8] R. S. Sutton and A. G. Barto. *Reinforcement learning: An introduction*. MIT Press, 1998.

[9] Herrnstein R.J. On the law of effect. *J. of the Exp. Anal. of Behav.*, 13(2):243–266, 1970.

[10] Dawson G.R. and Dickinson A. Performance on ratio and interval schedules with matched reinforcement rates. *Q. J. of Exp. Psych. B*, 42:225–239, 1990.

[11] Niv Y., Daw N.D., Joel D., and Dayan P. Motivational effects on behavior: Towards a reinforcement learning model of rates of responding. In *CoSyNe*, Salt Lake City, Utah, 2005.

[12] Aberman J.E. and Salamone J.D. Nucleus accumbens dopamine depletions make rats more sensitive to high ratio requirements but do not impair primary food reinforcement. *Neuroscience*, 92(2):545–552, 1999.

[13] Foster T.M., Blackman K.A., and Temple W. Open versus closed economies: performance of domestic hens under fixed-ratio schedules. *J. of the Exp. Anal. of Behav.*, 67:67–89, 1997.

[14] Staddon J.E.R. *Adaptive dynamics*. MIT Press, Cambridge, Mass., 2001.

[15] Mahadevan S. Average reward reinforcement learning: Foundations, algorithms and empirical results. *Machine Learning*, 22:1–38, 1996.

[16] Daw N.D., Kakade S., and Dayan P. Opponent interactions between serotonin and dopamine. *Neural Networks*, 15(4-6):603–616, 2002.

[17] Bolles R.C. *Theory of Motivation*. Harper & Row, 1967.

[18] Carr G.D. and White N.M. Effects of systemic and intracranial amphetamine injections on behavior in the open field: a detailed analysis. *Pharmacol. Biochem. Behav.*, 27:113–122, 1987.

[19] Jackson D.M., Anden N., and Dahlstrom A. A functional effect of dopamine in the nucleus accumbens and in some other dopamine-rich parts of the rat brain. *Psychopharmacologia*, 45:139–149, 1975.

[20] Dayan P. and Balleine B.W. Reward, motivation and reinforcement learning. *Neuron*, 36:285–298, 2002.

[21] McClure S.M., Daw N.D., and Montague P.R. A computational substrate for incentive salience. *Trends in Neurosc.*, 26(8):423–428, 2003.

[22] Daw N.D., Niv Y., and Dayan P. Uncertainty based competition between prefrontal and dorsolateral striatal systems for behavioral control. *Nature Neuroscience*, 8(12):1704–1711, 2005.

[23] Dickinson A., Balleine B., Watt A., Gonzalez F., and Boakes R.A. Motivational control after extended instrumental training. *Anim. Learn. and Behav.*, 23(2):197–206, 1995.

[24] Montague P.R., Dayan P., and Sejnowski T.J. A framework for mesencephalic dopamine systems based on predictive hebbian learning. *J. of Neurosci.*, 16(5):1936–1947, 1996.

[25] Satoh T., Nakai S., Sato T., and Kimura M. Correlated coding of motivation and outcome of decision by dopamine neurons. *J. of Neurosci.*, 23(30):9913–9923, 2003.

[26] Daw N.D., Courville A.C., and Touretzky D.S. Timing and partial observability in the dopamine system. In T.G. Dietterich, S. Becker, and Z. Ghahramani, editors, *NIPS*, volume 14, Cambridge, MA, 2002. MIT Press.

[27] Gallistel C.R. and Gibbon J. Time, rate and conditioning. *Psych. Rev.*, 107:289–344, 2000.
